# Doubly Hierarchical Geometric Representations for Strand-based Human Hairstyle Generation

**Yunlu Chen**[†*], **Francisco Vicente Carrasco**[†], **Christian Häne**[‡], **Giljoo Nam**[‡],
**Jean-Charles Bazin**[‡], **and Fernando De la Torre**[†]

[†]Carnegie Mellon University    [‡]Meta Reality Labs

## Abstract

We introduce a doubly hierarchical generative representation for strand-based 3D hairstyle geometry that progresses from coarse, low-pass filtered guide hair to densely populated hair strands rich in high-frequency details. We employ the Discrete Cosine Transform (DCT) to separate low-frequency structural curves from high-frequency curliness and noise, avoiding the Gibbs' oscillation issues associated with the standard Fourier transform in open curves. Unlike the guide hair sampled from the scalp UV map grids which may lose capturing details of the hairstyle in existing methods, our method samples optimal sparse guide strands by utilising $k$-medoids clustering centres from low-pass filtered dense strands, which more accurately retain the hairstyle's inherent characteristics. The proposed variational autoencoder-based generation network, with an architecture inspired by geometric deep learning and implicit neural representations, facilitates flexible, off-the-grid guide strand modelling and enables the completion of dense strands in any quantity and density, drawing on principles from implicit neural representations. Empirical evaluations confirm the capacity of the model to generate convincing guide hair and dense strands, complete with nuanced high-frequency details.[1]

## 1 Introduction

The quest for realistic virtual humans is a cornerstone of modern computer graphics, with high-quality 3D hair strand generation being one of its most intricate challenges. Our work focuses on the generation of strand hair with machine learning algorithms, which alleviates the labor-intensive process inherent in crafting digital hairstyles—a task that requires meticulous attention to detail and consumes a disproportionate amount of time and artistic resources. In addition, it strives to establish a generative prior, a foundational blueprint that not only streamlines the creation process but also enhances the reconstruction capabilities essential for virtual human applications.

Hair exhibits a wide spectrum of morphological details, from the subtlest of low-frequency principle directions and waves to the complexity of high-frequency curls, as described by the physics of elastic rods and Kirchhoff's theories [4, 22] on the dynamics of twisting filaments. These high-frequency elements often include curly helical structures [3, 15] as well as extraneous noise that can detract from accurately modelling the hair's principal growth direction. To counteract this, we advocate for a frequency decomposition approach to extract the principal direction as a low-frequency signal. This technique is vital in distilling the essence of the hair's natural trajectory, ensuring that the core path of growth is clearly defined and free from the visual clutter of high-frequency noise. Such a focus on low-frequency signals is not only important but also highly effective in generating hair models that

---

[*]Correspondence: `yunluchenxyz@gmail.com`.
[1]All data processing and use of models were conducted at CMU.

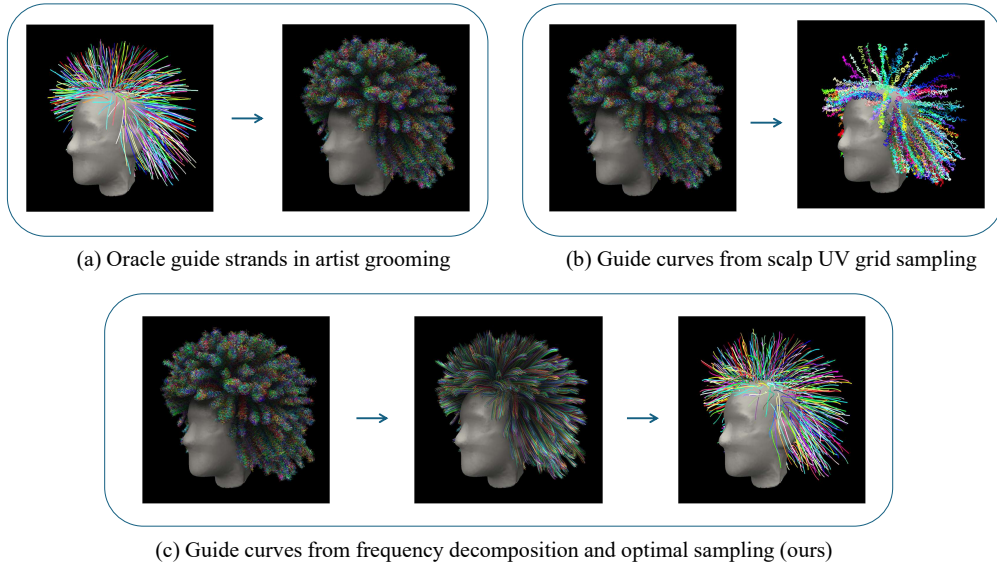

(a) Oracle guide strands in artist grooming  (b) Guide curves from scalp UV grid sampling

(c) Guide curves from frequency decomposition and optimal sampling (ours)

Figure 1: (a) The oracle guide curves from a grooming project in Blender [8] for principle hair growing directions. (b) Existing works simply sample guide curves from dense strands regularly on a UV grid of the scalp, which may contain unnecessary signals of high-frequency noise for neighbouring dense strands. (c) We model guide curves by frequency decomposition and $k$-medoids clustering for optimal sampling a subset of smooth guide hair strands, which are utilized in training our hierarchical hair generation model.

are true to the natural flow and inherent physical properties of hair, providing a robust basis for more nuanced and detailed simulations.

The complexity of hair data has led to the adoption of coarse guide strands as a foundational step in many hair modelling methods, reflecting the guide hair-based artist grooming in Fig. 1(a). These existing methods [32, 40, 48, 41], however, simplify the guide hair extraction to downsampling to a lower grid resolution of the 2D UV map of the head scalp, as in Fig. 1(b), which comes with some notable drawbacks. Firstly, simply extracting the full spectrum of each guide strand can be burdened by high-frequency details and noise irrelevant to adjacent strands, while the oracle guide strands are usually smooth curves for the hair's principal growing direction. Secondly, sampling from UV map grids can sometimes miss intricate hairstyle details and may produce guide strands that are less representative, as illustrated in Fig. 3. In addition, the grid sampling often results in a suboptimal density with less hair on the top of the head scalp but more hair on the side, since the UV mapping from a head scalp to the Euclidean $\mathbb{E}^2$ space is not area-preserving.

In response to these challenges, we introduce a novel doubly hierarchical representation of strand hair geometry, by introducing frequency decomposition and optimal sampling to extract guide hair, as illustrated in Fig. 1(c). This approach aligns more closely with contemporary computer graphics tools designed to craft artistically groomed hairstyles. From the perspective of learning, our design to first learn low-frequency components followed by high-frequency ones adheres to the well-known frequency principle or spectral bias [28] of generalisation in neural networks. Additionally, optimal sampling ensures consistency within the coarse-to-fine learning pipeline. By refining the guide curves this way, we aim to capture the essential form and structure of hairstyles more effectively, paving the way for more accurate and visually pleasing hair modelling.

We develop neural models to process our novel and sophisticated hierarchical representation of strand hair geometry, which encompasses a variety of data forms. To accommodate the flexible off-the-grid modelling of guide strands, our neural model architecture for generation of guide strands adapts permutation-equivariant geometric deep learning models [27, 45] that do not rely on a fixed feature grid within a Euclidean domain, with specific modifications tailored to the hair strand problem. Additionally, our densification model, designed to synthesize densely populated strands from sparse guides, draws inspiration from the continuous and resolution-free implicit neural representations [24] and graph message passing [13] techniques. This approach enables the model to effectively handle fine strands with varying numbers and densities.

To encapsulate our methodology and the contributions of our work:

- Our approach involves constructing a hierarchical generative model for 3D hair strands that begins with the formation of coarse guide curves through low-pass filtering and culminates in the generation of dense hair strands that incorporate intricate high-frequency details.
- We utilize the Discrete Cosine Transform (DCT) to obtain the coarse and low-pass filtered guide curves. This technique effectively distinguishes the fundamental shape of the hair from the more complex aspects of curliness and noise, thereby circumventing the Gibbs' oscillation problem that often plagues the standard Fourier transform when applied to open curves.
- The guide hair strands are sampled using $k$-medoids clustering centres from the dense hair data. This novel method is proven to be theoretically optimal in preserving the original hairstyle's features compared to the conventional practice of sampling from a regular 2D grid on a UV map.
- We adapt the family of non-Euclidean geometric deep learning models and develop a permutation-equivariant architecture for learning on hair strands, instead of 2D CNNs, for more flexible modelling of off-the-grid guide strands.
- We propose a novel neural mechanism for learning strand interpolation. Inspired by implicit neural representations and graph message passing, our method handles modelling any amount of dense strands at any sampling density, and enables end-to-end joint training with guide strands.

## 1.1 Related Work

Recent learning algorithms for strand-based hair focuses on capture and reconstruction from single [33, 43, 47] or multiple views [23, 32, 40], often relying on intermediate representations of volumetric occupancies [33, 46, 43, 40] or orientation fields from image gradient cues [33, 46, 40, 47, 43]. These approaches mostly have a focus on optimising strand growth or connecting segments, while learning applies to other intermediate representations (e.g. orientation field) but not directly on strands, thus not requiring hierarchical strand representation with abstraction.

Our work focuses on building a generative model directly on human hairstyles in the form of a set of strands, which has been an established representation in industry [2, 10], [38], owing to its compatibility with physics-based applications. Consequently, strands have emerged as a favoured representation in numerous computer vision, machine learning, and graphics projects. The most related works to our approach for strand hair generation are recent methods of GroomGen [48] and HAAR [41], both relying on off-the-shelf VAE codec representations of strands [32] mapped on the Euclidean domain of discretized UV map, optionally followed by a coarse-to-fine pipeline in accordance with different UV grid resolutions [48]. The early work from Wang et al. [42] focused on a slightly different task of exemplar-based synthesis of strand hair, which requires a base strand hairstyle or a combination of two for the global shape and synthesizes local textural details.

In contrast to existing strand-based hair generation methods [48, 41] that rely on hair representations from prior work [32], we introduce a novel doubly hierarchical hair representation and associated neural models. Our approach begins by extracting guide hairs from flexible root locations to achieve an optimal set of strands that faithfully preserve the original hairstyle through a non-Euclidean representation. Subsequently, we apply frequency decomposition using the Discrete Cosine Transform (DCT) to achieve a compact representation, eliminating the need for training off-the-shelf codecs and enabling end-to-end optimization. Compared to grid UV maps combined with strand codecs and 2D CNNs, our method is more flexible, sophisticated, and exhibits superior performance. Additionally, our grooming generation technique allows sampling of an arbitrary number of dense strands from any location, without being constrained by UV grid resolution.

## 2 Extracting the Hierarchy from Strand Hairstyles

A human's hair can be seen as a set of strands $\mathcal{H} = \{l_i\}_{i=1}^{N}$, where each strand is a 3D curve $l(t) : [0, 1] \mapsto \mathbb{R}^3$. In practice, strand $l$ is usually available as a polyline with discretely sampled control points in a sequence of $n$ 3D points: $l = [l(0), l(1), \ldots, l(n-1)] \in \mathbb{R}^{n \times 3}$ sampled from the continuous curve. The roots of the curves are attached to the head scalp $\mathcal{M}$: $l(0) \in \mathcal{M}$. In this work we consider all the hairstyles are aligned to the same human head scalp geometry.

We aim at a hierarchical modelling of human hairstyles due to the high complexity of the human hair data. Inspired by the guide strand-based modelling in artist strand hair creation, we propose to smoothen the strands with frequency decomposition and optimally sample coarse guide hair strands.

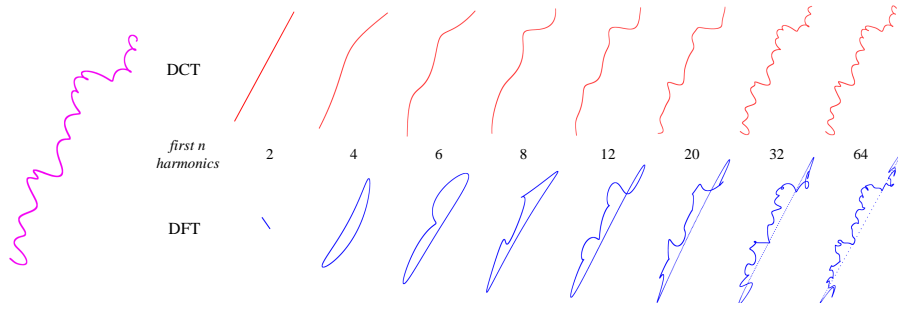

Figure 2: Comparing frequency-based smoothing of an exemplar curly strand. Smoothing from the DFT biases towards closed curves and suffers from Gibbs' oscillations [12] for hair strands as open curves. In contrast, The DCT has a strong energy compaction property and more flexible boundary conditions, facilitating encoding open curves with fewer harmonics.

## 2.1 Frequency decomposition of strands as open curves

Human hair exhibits a spectrum of morphologies, ranging from straightness to inherent curvature of planar waves or 3D helices from a confluence of the strand's physical attributes, such as mass, length, and tensile elasticity [22]. For undulating and helical strands, the predominant axis of growth can be conceptualized as a smooth, low-frequency trajectory, whereas the detailed twists and noise are considered high-frequency features. Artists creating hairstyles typically start by making smooth, basic guide strands and then add detailed waves, curls, and slight randomness [2, 8]. Notwithstanding, contemporary learning-based algorithms for guide strand modelling have largely overlooked this intrinsic frequency paradigm.

In our approach, we separate hair strands into low- and high-frequency components, aligning with traditional artist-driven hair modelling techniques that utilize smooth guide strands. Initially, we model the smoothed, low-frequency base of the strand, then we enhance it with high-frequency waves, curls, and localized noise. This method not only adheres to the spectral bias principle [28] but also streamlines the learning process.

The Discrete Fourier transform (DFT) is a widely applied method for frequency decomposition. However, the DFT assumes strong periodicity of the signal, thus suffers from the Gibbs' phenomenon [12] with significant oscillations when processing on open curves whose start points and endpoints do not coincide with each other [11, 9], as depicted in Fig. 2.

Therefore, we adopt the Discrete Cosine Transform (DCT) [1], a Fourier-related transform that is analogous to the Discrete Fourier Transform (DFT) but employs only real-valued cosine basis components. The DCT is extensively utilized in image, video, and audio signal compression [30] due to its superior energy compaction properties, which allow for more efficient representation of signals with fewer coefficients. Additionally, the DCT offers more flexible boundary conditions that mitigate wrap-around effects, thereby reducing artefacts such as those associated with the Gibbs phenomenon. These advantages make the DCT particularly well-suited for encoding open curves using fewer harmonics. Formally, we apply the DCT to transfer the hair curve $l(t)$ with $t = 1, \ldots, n$ into frequency domain $\tilde{l}(\tau)$ by

$$\mathrm{DCT}(l(t)) : \tilde{l}_X(\tau) = (\frac{2}{n})^{\frac{1}{2}} \sum_{t=0}^{n-1} w(t)\, l_X(t)\, \cos{[\frac{\pi}{2n}(2t+1)\tau]}, \quad \text{where } w(t) = \begin{cases} \frac{1}{\sqrt{2}} & \text{if } \tau = 0, \\ 1 & \text{otherwise.} \end{cases} \tag{1}$$

The corresponding inverse DCT is the inverse function $\mathrm{iDCT} = \mathrm{DCT}^{-1}$. $l_X$ and $\tilde{l}_X$ denote the signal component on the X-axis coordinate in spatial and frequency spaces respectively. The same process also applies to Y- and Z-axes.

Due to the even symmetry of the cosine function, the DCT is equivalent to the DFT operating on the symmetrically extended signal sequence of twice the length, while the sine components are cancelled out. The periodicity extension of the signal no longer introduces strong discontinuity, which facilitates modelling open curves, as corroborated by Fig. 6. We refer to the literature for details [30, 9].

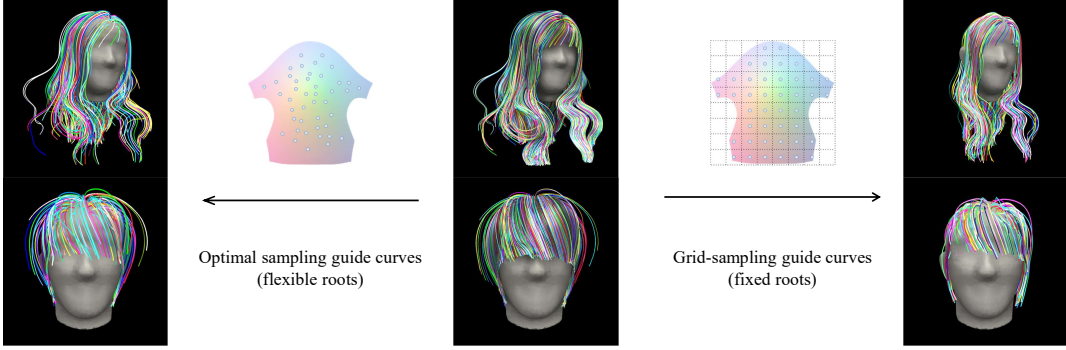

Figure 3: Comparison of guide hair curves extracted by $k$-medoids (left) and grid-sampling (right) from reference hairstyles (middle), both with an equal count of curves. It is evident that grid-sampling can omit crucial hairstyle details. In contrast, the off-the-grid optimal guide curve subset derived from our $k$-medoids method more accurately captures the essence and details of the original hairstyle.

## 2.2    Optimal sampling of coarse guide hair by $k$-medoids clustering

Learning-based approaches to strand-based hair modelling typically adopt a coarse-to-fine strategy, necessitating the generation of a set of coarse guide strands due to the computational demands of processing dense strands. This concept aligns with the traditional artist-driven hair creation workflow. Conventional methods predominantly employ a strategy of sampling guide strands by extracting those emanating from scalp locations situated at the grid centres of an unwrapped UV map of the scalp manifold. However, this regular grid sampling approach is suboptimal, often failing to capture certain strand clusters critical for detailing the hairstyle, as illustrated in Fig. 3. In contrast, we introduce a methodology for extracting a sparse subset of $k$ off-the-grid strands from the comprehensive dense hair strand set $\mathcal{H}^L = \{l^L\}$, with $l^L$ denoting low-pass filtered strand curves from the first $\tilde{n}^L$ DCT components as predominant growing directions, resulting in the optimal set $\mathcal{G}_k^*$ that more accurately mirrors the characteristics of the dense strands.

**Definition 1** (Optimal sampling guide hair curves)*. Given a set of dense low-pass filtered hair curves $\mathcal{H}^L$, the optimal sampling of coarse guide curve set $\mathcal{G}_k^*$ of cardinality $k$ is the subset of $\mathcal{H}^L$ which has the minimum possible bidirectional chamfer distance from $\mathcal{H}^L$. Formally, $\mathcal{G}_k^* = \arg\min_{\mathcal{G}_k \subset \mathcal{H}^L; |\mathcal{G}_k| = k} \mathtt{chamfer}(\mathcal{H}^L, \mathcal{G}_k)$*

Our investigation reveals that the $k$-medoids clustering method [16] delivers an optimal approach for identifying the optimal strand subset. $k$-medoids is a classical partitioning technique that organises data points into clusters with the aim of minimising the distance between the points within a cluster and a specific point within the same cluster, known as the medoid, which serves as the cluster's nucleus. Unlike the popular $k$-means algorithm which allows the cluster's centroid to be a virtual average of the data points, $k$-medoids selects actual data points as the medoids, enhancing the interpretability of the cluster centres. Formally, the objective of $k$-medoids is to partition the set $\mathcal{H}^L$ into a collection of disjoint clusters $\mathbb{S} = \{\mathcal{S}_1, \ldots, \mathcal{S}_k\}$, each with its corresponding medoids $\mathcal{U} = \{u_1, \ldots, u_k\}$ in such a way that the aggregate of dissimilarities to all elements within each cluster is minimised, as defined by the following objective:

$$\min_{\mathbb{S}} \sum_{i=1}^{k} \sum_{l^L \in \mathcal{S}_i} d(l^L, u_i), \quad \text{where } u_i = \arg\min_{l^L \in \mathcal{S}_i} \sum_{l'^L \in \mathcal{S}_i} d(l^L, l'^L), \tag{2}$$

and $d(\cdot, \cdot)$ is a dissimilarity measurement. This objective leads to the following observation:

**Theorem 1.** *The medoid set $\mathcal{U} = \{u_1, \ldots, u_k\}$ from the $k$-medoids clustering of $\mathcal{H}$ is the optimal sampled hair curve set $\mathcal{G}_k^*$, if aggregated squared Euclidean distance $d(l, l') = \frac{1}{n}\|l - l'\|_2^2 := \frac{1}{n}\sum_{t=0}^{n-1} \|l(t) - l'(t)\|_2^2$ for two individual curves is used as the divergence function for $k$-medoids.*

Theorem 1 establishes that the optimal strategy for sampling a specified number of guide strands from an original set of densely packed hair strands is to apply $k$-medoids clustering to the dense

strand set and select the resulting medoids as the guide strands. This resultant set of guide strands, termed the representative guide curve set in Definition 1, possesses the smallest possible Chamfer distance to the original dense strand set compared to any alternative sampling method employing the same number of strands.

Note that our hierarchical hair representation does not cluster dense strands as in [42] using the $k$-medoids method. Instead, we use only the resulting medoids to sample guide strands. This ensures optimal guide strand sampling by leveraging the derived medoid set. The reason we avoid using explicit clusters to represent dense hair is that the interaction between dense and sparse guide strands is better captured through sampling and interpolation. Each dense strand is influenced by multiple nearby guide strands rather than just one from a single cluster. Consequently, our guide strand representation and interpolation mechanism provide a more sophisticated and accurate approach than the cluster-based model in [42], resulting in a more nuanced depiction of hair structure.

## 2.3    Processing dense hair strands into hierarchy

We introduce the data structure of our doubly hierarchical hair data abstraction before it can be processed by neural models. We extract the low-frequency structure disentangled from high-frequency details for each strand as the principle growing directions, followed by optimally selecting a representative coarse subset of guide strands from the low-pass filtered dense hair data.

For each strand $l$, we map the root point on the scalp manifold $l(0) \in \mathcal{M}$ to its UV coordinate on the UV mapping of the scalp as $r \in \mathcal{M}_{\text{UV}} \subset \mathbb{R}^2$. We translate all strands to centralize the root points at the origin of the 3D space, and achieve the low-pass filtered curve with the first $\tilde{n}^L$ components in the DCT. We apply the DCT on the derivative $\dot{l}(t) = l(t+1) - l(t)$ instead of the original sequence of coordinates to ensure the same root position. To condense the signal, the reconstructed curve $l_i^L$ in the spatial domain is resampled to a reduced resolution of $2 \cdot \tilde{n}^L$ by fitting a cubic spline, adhering to the Nyquist sampling theorem [35] for precise discretisation. For high-frequency details, we maintain the signal in the form of DCT coefficients, denoted as $\tilde{l}^H \in \mathbb{R}^{(\tilde{n}^H - \tilde{n}^L) \times 3}$ A cut-off frequency $\tilde{n}^H$ is strategically selected to ensure compactness and exclusion of signal components exceeding $\tilde{n}^H$ as high-order noise. Practically, we set $\tilde{n}^L = 8$ and $\tilde{n}^H = 40$. Next, we employ $k$-medoids to identify the indices of the optimal coarse guides for each hairstyle, maintaining a consistent $k = 512$.

This way, we represent each strand as a tuple $(r, l^L, \tilde{l}^H)$, where $r$ is the root point's UV coordinate, $l^L$ is the low-pass filtered strand curve, and $\tilde{l}^H$ encapsulates the high-frequency details as DCT harmonics. Strands within the optimal coarse guide curve set are denoted as $(r^*, l^{L*})$ while high-frequency signals are excluded from guide curves. The learning process for hair generation is structured hierarchically: it commences with the generation of the guide curve set $\{(r^*, l^{L*})\}$, progresses to the densification of strands $\{(r, l^L)\}$, and culminates with the integration of high-frequency details $\{\tilde{l}^H\}$ when necessary, for hairstyles with heavy curliness or non-smoothness.

# 3    Learning to Generate Hierarchical Strand Hair

The generative process of our model is depicted in Fig. 4. We employ a variational autoencoder (VAE) [18, 31] to capture the distribution of hierarchical hair structures. The choice of VAE is motivated by its straightforwardness and proven effectiveness and efficiency in optimisation and sampling, while other categories of generative models are left to future exploration. The objective of a VAE is to learn a generative model $p_\theta(x, z) = p_\theta(z)p_\theta(x|z)$ for data $x$ and latent variables $z$. Since the true posterior is intractable, we approximate it using the latent encoder model $q_\phi(z|x)$ and optimise the variational lower bound (ELBO) on the marginal likelihood $p(x)$: $\mathcal{L}_{\text{VAE}} = \mathbb{E}_{q_\phi(z|x)}[\log p_\theta(x|z)] - \text{KL}(q_\phi(z|x)\|p_\theta(z))$, where the second term is the Kullback-Leibler divergence, quantifying the difference between the approximate and true posteriors.

## 3.1    Generation of the optimal coarse guide hair with dual-branch network

The guide VAE for the generation of the optimal guide curve set $\{(r^*, l^{L*})\}$ with a fixed cardinality $k$ is detailed in Fig. 4(b). The guide curves $\{(r^*, l^{L*})\}$ are conceptualized as a collection of data points, with the sequence $l^{L*}$ transformed to a feature vector $h^{L*} \in \mathbb{R}^{d_h L*}$ using a 1D convolutional encoder embedded on the domain of UV coordinates $r^* \in [0, 1]^2$. Our network architecture draws

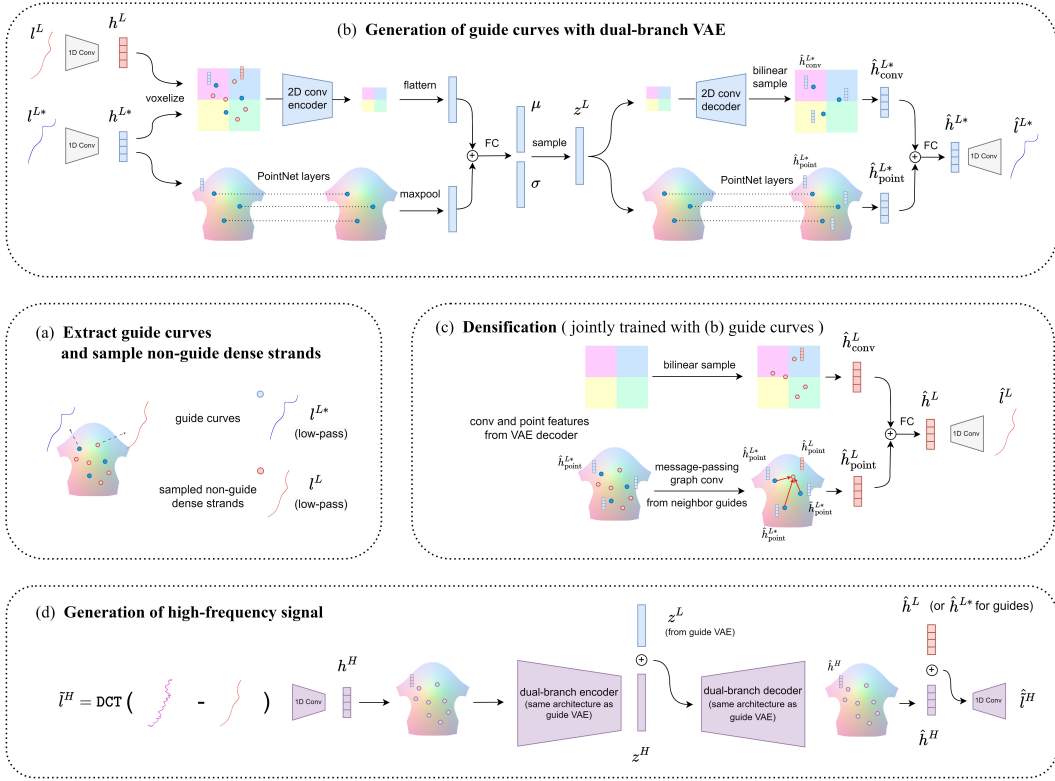

Figure 4: Generation pipeline overview: (a) During training, a small batch of dense strands and extracted guide curves are sampled from the UV map of the scalp to optimise the networks. (b) Guide curves are generated using a PVCNN VAE that incorporates 2D convolution and PointNet layers. (c) Dense strands are generated by aggregating features from the convolution grid via bilinear sampling and from neighbouring guide curves through graph convolution, with the densification module being jointly trained with the guide curve model. (d) High-frequency signals are refined using another dual-branch VAE, conditioned on the global and local latent features derived from the network responsible for generating the low-pass filtered principal strand signal.

inspiration from the Point-Voxel CNN (PVCNN) [19], leveraging its demonstrated efficiency and effectiveness in the domain of point set learning.

The architecture, akin to Point-Voxel CNN (PVCNN), utilizes a dual-branch approach to process point features: a PointNet branch and a convolution branch. Within the encoder, the point branch consists of a shared multilayer perceptron (MLP) that operates on all points, employing max pooling for aggregation as per the PointNet design [27]. The convolution branch, on the other hand, begins by partitioning the UV space $[0, 1]^2$ into a 2D grid of $W \times H$ resolution. This step is analogous to the "voxelisation" in PVCNN but is executed in 2D. Each grid cell is assigned an averaged feature vector derived from all data points whose root coordinates fall within it, yielding a feature tensor of dimensions $\mathbb{R}^{W \times H \times d_{h^{L*}}}$. This tensor is then processed by successive downsampling 2D convolution layers to distill a global feature. To enhance the feature representation in the encoder's convolution branch, we incorporate features from the sampled dense strands. These additional features are not utilized in the PointNet branch to avoid increasing the memory footprint, whereas the convolution branch feature is constrained to a fixed resolution. The mean and standard deviation for reparameterising the VAE latent code $z$ is the aggregated features from both branches.

The decoder's architecture mirrors that of the encoder. The point feature is bilinearly sampled from the feature grid at the end of the convolution branch, following the hybrid grid-implicit representations [26, 5]. The interpolated point feature is then reconstructed into the low-passed curve sequence $l^{L*}$.

**Sampling root positions from generated density map** The proposed dual-branch architecture learns generation of the curve sequence $p(\{l^{L*}\}|\{r^*\})$ given root positions $p(\{r^*\})$. Our objective extends to mastering the joint distribution $p(\{r^*, l^{L*}\})$, which we approach by decomposing $p(\{r^*\}, \{l^{L*}\})$ into $p(\{r^*\}) \cdot p(\{l^{L*}\}|\{r^*\})$ as optimising the root coordinates and curve sequence

jointly presents significant empirical challenges. While the dual-branch network is utilized for $p(\{l^{L*}\}|\{r^*\})$, learning $p(\{r^*\})$ could potentially be achieved with a distinct generative network, such as SetVAE [17]. Nevertheless, we employ a trick to generate a $W \times H$ grid map of root densities at the convolution branch decoder, achieved by tallying the roots within each grid cell. During inference, $p(\{r^*\})$ is derived from this grid density map, before the sequence signal $\{l^{L*}\}|\{r^*\}$ is generated. Details of the density map sampling are provided in the Appendix D. Although the density map sampling does not guarantee the exact same original root positions, empirically with a reasonable resolution of density map, we find that the resulting root positions correctly resemble the distribution of root points.

## 3.2 Densification

To accommodate our innovative off-the-grid guide curve design, we introduce a densification module inspired by the concept of implicit neural representations [24, 21, 7, 39] This approach enables the generation of an arbitrary number of fine strands, independent of the scalp UV map's resolution, by leveraging hybrid grid [26, 5] and graph [6] representations. Given a randomly sampled query root position $r_i \in \mathcal{M}_{\text{UV}}$, our goal is to derive fine strand features that align with the guide curve features within the VAE decoder. Features from the convolution branch $\hat{h}^{L*}_{\text{conv},j}$ are bilinearly sampled from the convolution branch, while for the PointNet branch, the query aggregates point-wise features $\hat{h}^{L*}_{\text{point},j}$ from the query's neighbouring guide curve locations $r^*_j \in \mathcal{N}_{r_i}$, employing a graph-hybrid implicit representation strategy [6] This method considers guide curves as anchor points and propagates signals to arbitrary query locations through message-passing graph convolution [13]: $\hat{h}^{L}_{\text{point},i} = \frac{1}{|\mathcal{N}_{r_i}|} \sum_j w_j(\text{MLP}(r_i, r^*_j - r_i, \hat{h}^{L*}_{\text{point},j}) + \hat{h}^{L*}_{\text{point},j})$ with a two-layer MLP, and

$w_j = \frac{1}{\|r^*_j - r_i\|_2} / \sum_{j'} \frac{1}{\|r^*_{j'} - r_i\|_2}$ is a decaying weight. The resulted features are passed to the same 1D convolution decoder in the guide curve VAE model.

Note that the generation of the optimal guide curves and sampled non-guide strands is trained jointly. This pipeline is designed to be invariant to the sampling density, allowing for the generation of any desired number of dense strands during inference. This way, learning of densification can be regarded as in an auto-decoder learning scheme [24] conditioned on the coarse guide strands, and the information and varieties of dense strands is stored in the latent code and the network.

## 3.3 Refinement of high-frequency details

Though the majority of hair samples in our dataset can be accurately represented by low-frequency signals, certain hairstyles exhibit high-frequency structures of curliness and noise. The model for learning the high-frequency components $\{\tilde{l}^H\}$ is a conditioned VAE with the same dual-branch architecture as the model that generates guide curves, except that the model is conditioned by concatenating the global latent from the guide model and the local dense strand feature. This way, we can sample different forms of high-frequency details of noise and curly structures from the same generated low-frequency components of strands.

## 4 Experiments

**Data** Our method is trained on a dataset comprising 658 synthetic strand hairstyles. Detailed information on the dataset and training procedures is available in the appendix.

**Frequency decomposition and guide curve extraction** Our evaluation focuses on the design choices made in modelling the hierarchical structure of hairstyles. As depicted in Fig. 6, we apply low-pass filtering with a designated cutoff frequency to the strands, either on the original coordinate sequence $l(t)$ or its derivative $\dot{l}(t)$, the latter being the approach we adopt. In both scenarios, the Discrete Cosine Transform (DCT) outperforms the Discrete Fourier Transform (DFT) in reconstructing the signal within the same low-pass frequency band. The derivative form of the signal, being simpler to model, enhances the performance of both DFT and DCT, with DCT showing superior effectiveness. Furthermore, in Table 1, we conduct a comparative analysis of the $k$-medoids method against baselines for guide curve extraction, namely grid-based sampling, farthest point

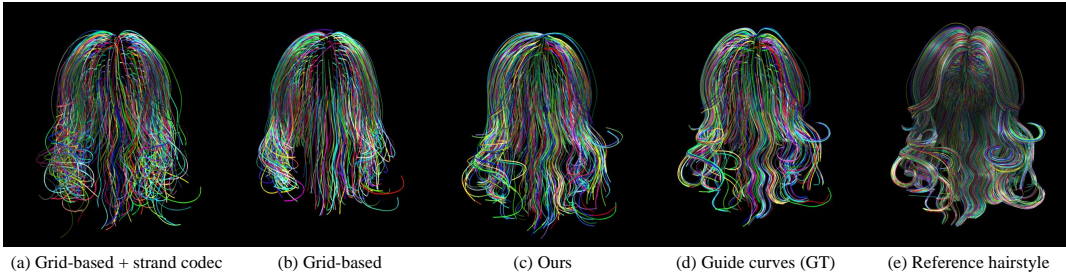

(a) Grid-based + strand codec   (b) Grid-based   (c) Ours   (d) Guide curves (GT)   (e) Reference hairstyle

Figure 5: The VAE reconstruction of guide curves, showcasing our method's enhanced performance due to informative guide curves and enriched network representation.

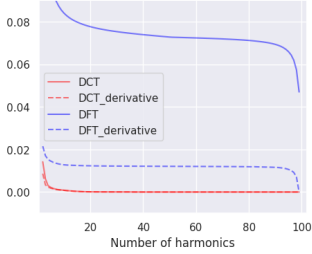

Figure 6: Euclidean distance between low-pass filtered and original strands.

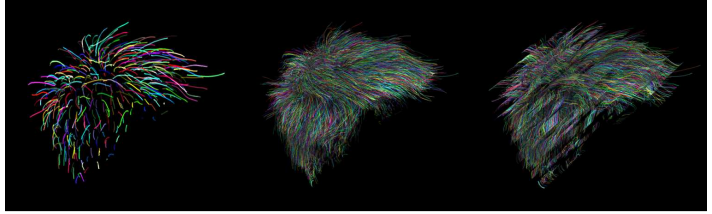

(a) Generated guide curves   (b) Our densification   (c) Nearest-neighbour upsample

Figure 7: Dense strands from VAE generated guide curves, with our model yielding natural-looking hair, in contrast to the nearest neighbour baseline which displays artefacts.

sampling (FPS), and $k$-means followed by projection of clustering centres to valid strands on the head scalp. The $k$-medoids approach demonstrably achieves a closer approximation to the dense strand set, as evidenced by the lower bidirectional chamfer distance (CD). These empirical findings align with our theoretical expectations, underscoring the efficacy of our methodological choices in capturing the intricate details of hair structure and texture.

**Reconstruction of guide curves with VAE** Our generative hair representation's efficacy is assessed using a set of 30 test examples, as detailed in Table 2 and illustrated in Fig. 5. Our method outperforms traditional grid-based hair representations, which map strands onto a scalp UV grid map and process them with purely 2D convolutional layers. The enhanced performance can be attributed to the rich representation derived from guide curves and our off-the-grid architectural approach. Direct comparison with closely related works such as Groom-Gen [48] and HAAR [41] is challenging due to lack of suitable evaluation metrics and the fact that they are trained on different datasets. Instead, we benchmark against a conceptually similar baseline representation: a grid-based Euclidean feature representation on the scalp UV-map, paired with a pretrained off-the-shelf strand VAE [32], which is the hair representation adopted by the prior work [41, 48]. This representation, however, falls short in performance. We posit that a strand codec VAE, originally designed as a generative prior for multiview hair reconstruction, may not be optimal for hairstyle-level generative tasks. By jointly optimising the strand encoder and decoder, our model circumvents the error accumulation inherent in the strand VAE approach, leading to more accurate and realistic hair generation.

Table 1: Evaluation on extracted guide curves (CD $(\times 10^{-4})$ to reference hair).

| Grid-sample | FPS | $k$-means + projection | $k$-medoids |
|---|---|---|---|
| 0.683 | 0.139 | 0.116 | **0.101** |

Table 2: Evaluating reconstruction of guide curves.

| CD $(\times 10^{-4})$ to: | Grid-based | Grid-based + strand codec | Ours |
|---|---|---|---|
| Input sampled strands | 1.521 | 1.597 | **0.906** |
| GT dense hair | 9.090 | 10.843 | **8.079** |

Table 3: Evaluation of densification on VAE reconstructed guide curves.

| | Nearest-neighbour | Ours |
|---|---|---|
| CD $(\times 10^{-4})$ to GT dense hair | 0.311 | **0.209** |

**Densification** Building on the guide reconstruction experiments, we further assess the quality of densified strands against the reference hair, as quantified in Table 3. Our method outperforms the baseline approach, which merely duplicates strands from the closest guide curve. Notably, our model demonstrates the ability to generate high-quality dense strands informed by the generated coarse

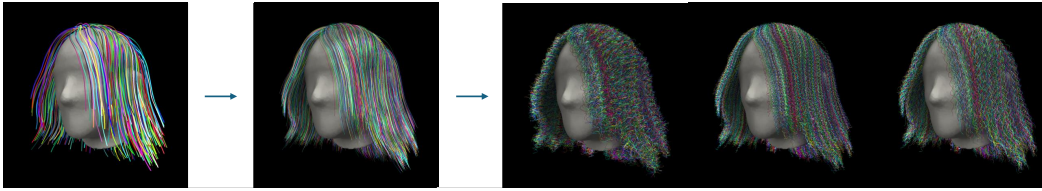

Figure 8: Generating diverse high-frequency details: Our high-frequency conditional VAE adds varied curliness and noise to hairstyles with guide curves and dense strands in low-frequency, demonstrating our model's ability to disentangle low- and high-frequency strand geometry.

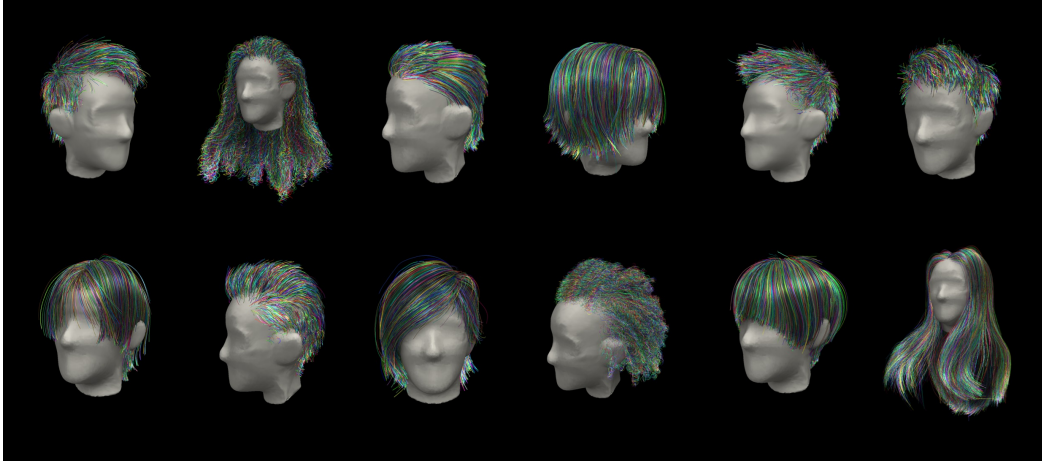

Figure 9: Strand-based hairstyles generated using our approach.

guide priors, benefiting from the joint training with the guide curve VAE. As illustrated in Fig. 7, our model is capable of refining a set of VAE-generated guide curves into dense strands with fine textural details, in contrast to the nearest neighbour upsampling method, which introduces unnatural artefacts.

**Creating varied high-frequency details in hair** Our methodology extends to generating diverse high-frequency details for each hairstyle, building upon the foundation of guide curves and dense strands in low-frequency created by our model. By employing a high-frequency conditional VAE, we introduce a variety of high-frequency details, such as curliness and noise, into the hair strands. This process is vividly illustrated in Fig. 8, which showcases the capability of our approach to effectively disentangle and separately model the components of hair strand signals in both low- and high-frequency domains. This disentanglement allows for the nuanced recreation of hair textures, enhancing the realism and diversity of the generated hairstyles.

**Additional results** Some outcomes of our hair generation pipeline are showcased in Fig. 9. Additional experiments and ablation results are conducted and reported in the appendix.

## 5   Conclusion

We propose a doubly hierarchical generative representation for strand that captures the full spectrum of hair details from low-frequency shapes to dense and fine details. By leveraging frequency decomposition and optimal sampling, our model surpasses traditional grid representations in preserving the authenticity of hairstyles. Our model design ensures the generation of flexible hair strands in any amount and density that are both diverse and realistic, free from the constraints of grid-based systems. Future work could explore the integration of dynamic hair behaviours and the adaptation of our model to generate a wider variety of hair types, further enhancing the realism and applicability of virtual hair in diverse digital environments.

**Acknowledgement**    We thank Georgina Linuesa Gomez for guidance on the knowledge of human hair types and artist grooming, Kai Qiu and Parth Nilesh for technical support, and anonymous reviewers for their valuable feedback. All data processing and use of models was conducted at CMU.

# References

[1] Nasir Ahmed, T_ Natarajan, and Kamisetty R Rao. Discrete cosine transform. *IEEE transactions on Computers*, 100(1):90–93, 1974.

[2] Autodesk, INC. Maya.

[3] Florence Bertails, Basile Audoly, Marie-Paule Cani, Bernard Querleux, Frédéric Leroy, and Jean-Luc Lévêque. Super-helices for predicting the dynamics of natural hair. *ACM Transactions on Graphics (TOG)*, 25(3):1180–1187, 2006.

[4] Florence Bertails, Basile Audoly, Bernard Querleux, Frédéric Leroy, Jean-Luc Lévêque, and Marie-Paule Cani. Predicting natural hair shapes by solving the statics of flexible rods. In *Eurographics short papers*. Eurographics, 2005.

[5] Eric R Chan, Connor Z Lin, Matthew A Chan, Koki Nagano, Boxiao Pan, Shalini De Mello, Orazio Gallo, Leonidas J Guibas, Jonathan Tremblay, Sameh Khamis, et al. Efficient geometry-aware 3d generative adversarial networks. In *Proceedings of the IEEE/CVF conference on computer vision and pattern recognition*, pages 16123–16133, 2022.

[6] Yunlu Chen, Basura Fernando, Hakan Bilen, Matthias Nießner, and Efstratios Gavves. 3d equivariant graph implicit functions. In *Computer Vision–ECCV 2022: 17th European Conference, Tel Aviv, Israel, October 23–27, 2022, Proceedings, Part III*, pages 485–502. Springer, 2022.

[7] Zhiqin Chen and Hao Zhang. Learning implicit fields for generative shape modeling. In *Proceedings of the IEEE Conference on Computer Vision and Pattern Recognition*, pages 5939–5948, 2019.

[8] Blender Online Community. *Blender - a 3D modelling and rendering package*. Blender Foundation, Stichting Blender Foundation, Amsterdam, 2018.

[9] Cyril H Dommergues, Jean-Louis Dommergues, and Eric P Verrecchia. The discrete cosine transform, a fourier-related method for morphometric analysis of open contours. *Mathematical Geology*, 39:749–763, 2007.

[10] Epic Games. Unreal engine.

[11] Gerald B Folland. *Fourier analysis and its applications*, volume 4. American Mathematical Soc., 2009.

[12] J Willard Gibbs. Fourier's series. *Nature*, 59.1539, 1899.

[13] Justin Gilmer, Samuel S Schoenholz, Patrick F Riley, Oriol Vinyals, and George E Dahl. Neural message passing for quantum chemistry. In *International conference on machine learning*, pages 1263–1272. PMLR, 2017.

[14] Kaiming He, Xiangyu Zhang, Shaoqing Ren, and Jian Sun. Deep residual learning for image recognition. In *Proceedings of the IEEE conference on computer vision and pattern recognition*, pages 770–778, 2016.

[15] Liwen Hu, Chongyang Ma, Linjie Luo, and Hao Li. Single-view hair modeling using a hairstyle database. *ACM Transactions on Graphics (ToG)*, 34(4):1–9, 2015.

[16] Leonard Kaufman. Partitioning around medoids (program pam). *Finding groups in data*, 344:68–125, 1990.

[17] Jinwoo Kim, Jaehoon Yoo, Juho Lee, and Seunghoon Hong. Setvae: Learning hierarchical composition for generative modeling of set-structured data. In *Proceedings of the IEEE/CVF Conference on Computer Vision and Pattern Recognition*, pages 15059–15068, 2021.

[18] Diederik P. Kingma and Max Welling. Auto-encoding variational bayes. In Yoshua Bengio and Yann LeCun, editors, *ICLR*, 2014.

[19] Zhijian Liu, Haotian Tang, Yujun Lin, and Song Han. Point-voxel cnn for efficient 3d deep learning. *Advances in neural information processing systems*, 32, 2019.

[20] Ilya Loshchilov and Frank Hutter. Decoupled weight decay regularization. *arXiv preprint arXiv:1711.05101*, 2017.

[21] Lars Mescheder, Michael Oechsle, Michael Niemeyer, Sebastian Nowozin, and Andreas Geiger. Occupancy networks: Learning 3d reconstruction in function space. In *Proceedings of the IEEE Conference on Computer Vision and Pattern Recognition*, pages 4460–4470, 2019.

[22] Jay T Miller, Arnaud Lazarus, Basime Audoly, and Pedro M Reis. Shapes of a suspended curly hair. *Physical review letters*, 112(6):068103, 2014.

[23] Giljoo Nam, Chenglei Wu, Min H Kim, and Yaser Sheikh. Strand-accurate multi-view hair capture. In *Proceedings of the IEEE/CVF Conference on Computer Vision and Pattern Recognition*, pages 155–164, 2019.

[24] Jeong Joon Park, Peter Florence, Julian Straub, Richard Newcombe, and Steven Lovegrove. Deepsdf: Learning continuous signed distance functions for shape representation. In *Proceedings of the IEEE Conference on Computer Vision and Pattern Recognition*, pages 165–174, 2019.

[25] Adam Paszke, Sam Gross, Francisco Massa, Adam Lerer, James Bradbury, Gregory Chanan, Trevor Killeen, Zeming Lin, Natalia Gimelshein, Luca Antiga, et al. Pytorch: An imperative style, high-performance deep learning library. *arXiv preprint arXiv:1912.01703*, 2019.

[26] Songyou Peng, Michael Niemeyer, Lars Mescheder, Marc Pollefeys, and Andreas Geiger. Convolutional occupancy networks. In *Computer Vision–ECCV 2020: 16th European Conference, Glasgow, UK, August 23–28, 2020, Proceedings, Part III 16*, pages 523–540. Springer, 2020.

[27] Charles R Qi, Hao Su, Kaichun Mo, and Leonidas J Guibas. Pointnet: Deep learning on point sets for 3d classification and segmentation. In *Proceedings of the IEEE conference on computer vision and pattern recognition*, pages 652–660, 2017.

[28] Nasim Rahaman, Aristide Baratin, Devansh Arpit, Felix Draxler, Min Lin, Fred Hamprecht, Yoshua Bengio, and Aaron Courville. On the spectral bias of neural networks. In *International Conference on Machine Learning*, pages 5301–5310. PMLR, 2019.

[29] Prajit Ramachandran, Barret Zoph, and Quoc V Le. Searching for activation functions. *arXiv preprint arXiv:1710.05941*, 2017.

[30] K Ramamohan Rao and Ping Yip. *Discrete cosine transform: algorithms, advantages, applications*. Academic press, 2014.

[31] Danilo Jimenez Rezende, Shakir Mohamed, and Daan Wierstra. Stochastic backpropagation and variational inference in deep latent gaussian models. In *International conference on machine learning*, volume 2, page 2, 2014.

[32] Radu Alexandru Rosu, Shunsuke Saito, Ziyan Wang, Chenglei Wu, Sven Behnke, and Giljoo Nam. Neural strands: Learning hair geometry and appearance from multi-view images. In *European Conference on Computer Vision*, pages 73–89. Springer, 2022.

[33] Shunsuke Saito, Liwen Hu, Chongyang Ma, Hikaru Ibayashi, Linjie Luo, and Hao Li. 3d hair synthesis using volumetric variational autoencoders. *ACM Transactions on Graphics (TOG)*, 37(6):1–12, 2018.

[34] Erich Schubert and Lars Lenssen. Fast k-medoids clustering in rust and python. *Journal of Open Source Software*, 7(75):4183, 2022.

[35] Claude E Shannon. Communication in the presence of noise. *Proceedings of the IRE*, 37(1):10–21, 1949.

[36] Yuefan Shen, Shunsuke Saito, Ziyan Wang, Olivier Maury, Chenglei Wu, Jessica Hodgins, Youyi Zheng, and Giljoo Nam. Ct2hair: High-fidelity 3d hair modeling using computed tomography. *ACM Transactions on Graphics (TOG)*, 42(4):1–13, 2023.

[37] Wenzhe Shi, Jose Caballero, Ferenc Huszár, Johannes Totz, Andrew P Aitken, Rob Bishop, Daniel Rueckert, and Zehan Wang. Real-time single image and video super-resolution using an efficient sub-pixel convolutional neural network. In *Proceedings of the IEEE conference on computer vision and pattern recognition*, pages 1874–1883, 2016.

[38] SideFX. Houdini.

[39] Vincent Sitzmann, Julien Martel, Alexander Bergman, David Lindell, and Gordon Wetzstein. Implicit neural representations with periodic activation functions. *Advances in Neural Information Processing Systems*, 33, 2020.

[40] Vanessa Sklyarova, Jenya Chelishev, Andreea Dogaru, Igor Medvedev, Victor Lempitsky, and Egor Zakharov. Neural haircut: Prior-guided strand-based hair reconstruction. In *Proceedings of the IEEE/CVF International Conference on Computer Vision*, pages 19762–19773, 2023.

[41] Vanessa Sklyarova, Egor Zakharov, Otmar Hilliges, Michael J Black, and Justus Thies. Haar: Text-conditioned generative model of 3d strand-based human hairstyles. *arXiv preprint arXiv:2312.11666*, 2023.

[42] Lvdi Wang, Yizhou Yu, Kun Zhou, and Baining Guo. Example-based hair geometry synthesis. In *ACM SIGGRAPH 2009 papers*, pages 1–9, 2009.

[43] Keyu Wu, Yifan Ye, Lingchen Yang, Hongbo Fu, Kun Zhou, and Youyi Zheng. Neuralhdhair: Automatic high-fidelity hair modeling from a single image using implicit neural representations. In *Proceedings of the IEEE/CVF Conference on Computer Vision and Pattern Recognition*, pages 1526–1535, 2022.

[44] Yuxin Wu and Kaiming He. Group normalization. In *Proceedings of the European conference on computer vision (ECCV)*, pages 3–19, 2018.

[45] Manzil Zaheer, Satwik Kottur, Siamak Ravanbakhsh, Barnabas Poczos, Ruslan Salakhutdinov, and Alexander Smola. Deep sets. *arXiv preprint arXiv:1703.06114*, 2017.

[46] Meng Zhang and Youyi Zheng. Hair-gan: Recovering 3d hair structure from a single image using generative adversarial networks. *Visual Informatics*, 3(2):102–112, 2019.

[47] Yujian Zheng, Zirong Jin, Moran Li, Haibin Huang, Chongyang Ma, Shuguang Cui, and Xiaoguang Han. Hairstep: Transfer synthetic to real using strand and depth maps for single-view 3d hair modeling. In *Proceedings of the IEEE/CVF Conference on Computer Vision and Pattern Recognition*, pages 12726–12735, 2023.

[48] Yuxiao Zhou, Menglei Chai, Alessandro Pepe, Markus Gross, and Thabo Beeler. Groomgen: A high-quality generative hair model using hierarchical latent representations. *ACM Transactions on Graphics (TOG)*, 42(6):1–16, 2023.

## A  Border Impacts

Advancements in hair generation technology, such as our work, can enrich virtual realism and inclusivity, benefiting industries like gaming and entertainment. However, they also raise concerns about the potential misuse in Deepfakes and the reinforcement of unrealistic beauty standards. Balancing these innovations with ethical considerations is essential to ensure their positive impact.

## B  Proof of Theorem 1

Assume that from the $k$-medoids algorithm, we obtain the set of medoids $\mathcal{U} = \{u_1, \ldots, u_k\}$ with each $u_i$ from the set of dense hair strands $\mathcal{H}$. Then, from Eq. (2), $\mathcal{U}$ achieves minimum sum of cluster element-to-medoid distance $\sum_{i=1}^{k} \sum_{l^L \in \mathcal{S}_i} d(l^L, u_i)$.

Next, from the algorithm implementation, each element $l^L$ in $\mathcal{H}$ is closest (or equally close) to the medoid of its own cluster than that of any other cluster, so $\mathcal{U}$ is the subset of $\mathcal{H}$ with cardinality $k$ that achieves minimum $\sum_{i=1}^{k} \sum_{l^L : \min_{u_i \in \mathcal{U}} d(l^L, u_i)} d(l^L, u_i) = \sum_{l^L \in \mathcal{H}} \min_{u_i \in \mathcal{U}} d(l^L, u_i)$ which is the minimum sum of each dense strand $l^L$ with its nearest medoid $u$. After taking the average (divided by a constant $|\mathcal{H}|$), $\frac{1}{\mathcal{H}} \sum_{l^L \in \mathcal{H}} \min_{u_i \in \mathcal{U}} d(l^L, u_i)$ is in the form of a **unidirectional chamfer distance** from $\mathcal{H}$ to $\mathcal{U}$. So $\mathcal{U}$ achieves the minimum unidirectional chamfer distance from $\mathcal{H}$, from all possible subset of $\mathcal{H}$ with cardinality $k$.

Then we show that in the reverse direction, the unidirectional chamfer distance from $\mathcal{U}$ to $\mathcal{H}$, $\frac{1}{|\mathcal{U}|} \sum_{u_i \in \mathcal{U}} \min_{l^L \in \mathcal{H}} d(l^L, u_i)$, is constantly 0. This is easy to infer because $\mathcal{U}$ is a subset of $\mathcal{H}$, and each $u_i$ can find the same element from $\mathcal{H}$ that is closest to itself with distance 0. Aggregating both directions, we conclude that $\mathcal{U}$, from all possible subset of $\mathcal{H}$ with cardinality $k$, achieves the minimum (bidirectional) chamfer distance between $\mathcal{U}$ and $\mathcal{H}$, i.e., $\mathcal{U}$ is the optimally sampled subset of $\mathcal{H}$ with cardinality $k$ according to Definition 1.

## C  Additional Experiments

**Ablation on designing choices**  We report ablation results to verify the design choices in our approach in Tables 4 and 5.

In Table 4, we assess the VAE reconstruction efficacy with various methodological alternatives. Opting for randomly sampled guides over $k$-medoids derived ones compromises representativeness, adversely affecting performance. Utilizing frequency space coefficients to represent principal low-pass curves also leads to increased error, justifying our choice to revert low-pass filtered signals to the spatial domain. Our integrated training approach for both strand and hairstyle levels proves superior to employing a pretrained strand VAE codec, as utilized in other studies.

We also explored substituting graph aggregation with the nearest guide's PointNet branch feature for each dense strand's query root position, which negatively impacted the upsampled strands' accuracy.

For the high-frequency model, we benchmarked against using a pretrained strand codec for reconstructing each strand's high-frequency signal. The mean average error in reconstructing high-frequency coefficients suggests that an end-to-end optimisation of strand features is more effective than relying on a pretrained codec.

**Ablation on the frequency threshold for the low-pass filtered hair curves**  we expect frequency decomposition helps the representation quality in the neural network model as inspired by the spectral bias principle [28]. We show the additional ablation experiments on varying frequency threshold in reconstructing straight and curly hair strands with both low- and high-frequency details in Table 6, with hair strands reconstructed by aggregating results from both our low- and high-frequency models. We observe that the frequency threshold in the range from 8 to 12 is optimal, and empirically we use 8 which is more efficient. When the frequency threshold is too low, the low-filtered signal does not capture enough information of the principle growing direction. And when the frequency threshold is too high, high-frequency structure cannot be encoded more efficiently by DCT coefficients, and the increased computation cost hinders optimisation. Our representation makes use of both spatial and spectral domain with correct setup of frequency threshold.

Table 4: Ablation on VAE reconstruction of low-freq. guide curves and densification (CD $\times 10^4$ $\downarrow$).

|  | guide-to-GT guide | guide-to-GT dense | dense-to-GT dense |
|---|---|---|---|
| Ours | 0.906 | 8.079 | 0.209 |
| Randomly sampled strands as guide | 0.997 | 8.220 | 0.213 |
| $l^L \to \tilde{l}^L$ (frequency space) | 1.152 | 8.304 | 0.226 |
| With off-the-shelf strand codec | 0.966 | 8.117 | 0.216 |
| W/o graph aggregation in densification | 0.913 | 8.102 | 0.214 |

Table 5: Ablation on high-freq. reconstruction (MAE $\downarrow$).

|  | MAE $\downarrow$ |
|---|---|
| Ours | 0.137 |
| With off-the-shelf strand codec (freq. space) | 0.145 |

Table 6: Ablation on varying frequency threshold (VAE reconstruction)

| Frequency threshold | 4 | 6 | 8 | 10 | 12 | 16 | 20 | 32 | 64 | 100 |
|---|---|---|---|---|---|---|---|---|---|---|
| CD $(\times 10^{-4})$ | 1.494 | 1.122 | 1.010 | 1.014 | 1.009 | 1.024 | 1.188 | 1.432 | 1.865 | 1.849 |

**Subjective evaluation**  we conduct user study with human evaluation to compare our method with the unconditional model of HAAR [41], which is the only open-source generative model for strand hair at the time of the rebuttal period of NeurIPS 2024. We randomly generate 30 hairstyles using our method without selection, and 30 from HAAR, in total 60 examples, randomly shuffled before presented to the users. Each user will give a score 1-10 to each hairstyle on how realistic the generated hairstyle looks. We provide examples and instructions on the Google form in Fig. 10. Users participate in the experiments voluntarily, and each user takes 5 to 10 minutes to finish the experiment. We collected 54 valid responses. The resulting average scores are in Table 7. suggesting the advantage of our generation over HAAR. We also show results in different hairstyles categories, For short hair, both methods perform good. Our method perform significantly better on long hair and especially curly hair, due to our sophisticated representation design, e.g. frequency decomposition, the learned neural interpolation, end-to-end training that facilitate optimisation.

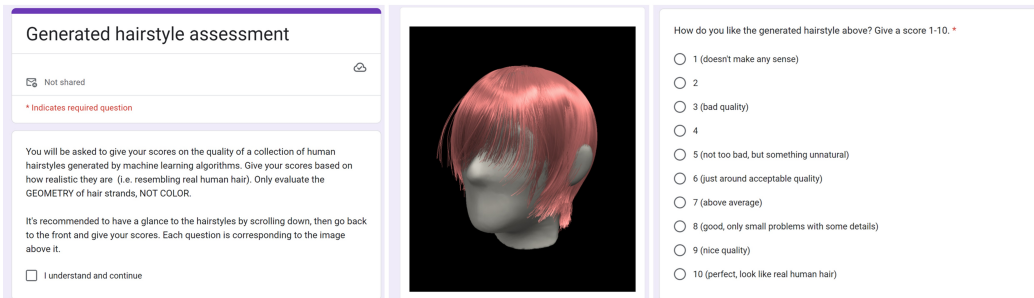

Figure 10: Screenshots of instructions on the Google form, with an example of generated hairstyle (1/60) in the middle. We rendered hair in the same color and not from black, brown or blonde to avoid racial bias.

Table 7: User assessment of hair generation quality. Subjective rating of scores from 1 to 10

|  | HAAR | ours |
|---|---|---|
| All hairstyles | $6.106 \pm 0.254$ | $\mathbf{6.933} \pm 0.237$ |
| Short non-curly | $7.133 \pm 0.287$ | $\mathbf{7.378} \pm 0.254$ |
| Long non-curly | $5.811 \pm 0.372$ | $\mathbf{6.456} \pm 0.328$ |
| Curly | $4.933 \pm 0.424$ | $\mathbf{6.656} \pm 0.347$ |

# D   Implementation Details

Our method is implemented with PyTorch [25]. All experiments are conducted on a single Nvidia RTX A4500 GPU. We use AdamW [20] with a learning rate of $3 \times 10^{-4}$ for 100k iterations for both low-frequency and high-frequency models, with an exponential learning rate decay to $3 \times 10^{-6}$ at the end of training. The batch size is 32.

During training, we apply random horizontal flipping and online scale augmentation in a range of [0.95, 1.1]. To prevent optimisation of the network from overfitting, in training we oversample 1024 guide curves in data preprocessing and further subsample 512 of them in training, while half of the guide set can still capture the characteristics of the hairstyle. We also sample another batch of 512 non-guide strands in each hairstyle in training to enrich the convolution branch representation. During evaluation we are able to generate arbitrary number of strands but we limit the number to 10000 for visualizations in the paper to match the dataset. The downsampling and upsampling of strands is achieved by fitting cubic splines with corresponding numbers of control points.

We use the visualization tool provided by the code repository in [36] for our work. We use the code from [34] to extract $k$-medoids.

**The 1D convolution encoder and decoder**   The 1D conv encoder applied on each low-frequency curve $l^L \in \mathbb{R}^{\tilde{n}^L \times 3}$ consists of two downsampling blocks to map the input sequence resolution from 16 to 8 and 8 to 4. Each downsampling block consists of four 1D conv layers with kernel size 3, and every third conv layer of each block applies a stride of 2 for downsampling. In between every two conv layers we add a residual skip connection [14] and if downsampling is involved then the skip connection is replaced by a 1D conv of kernel size 1 with a stride of 2. The dimension of the latent feature is 16. After the sequence length becomes 4 we flatten the sequence data to a vector from $4 \times 16$ to 64. Then we attach two fully connected layers to map each layers to map the latent dimension to 32. The 1D conv decoder is in a reversed architecture of the encoder except that the upsampling conv layer is in a depth-to-space design [37].

**The 2D convolution and PointNet branches**   After each strand sequence is transformed into a vector code, the 2D conv branch encoder aggregates them to a grid of $W \times H = 64 \times 64$ resolution, and downsampled to $2 \times 2$ after five downsampling blocks. The downsampling blocks are similar to those in 1D but with $3 \times 3$ kernels. And the latent dimension in downsampling blocks are 32, 64, 128, and 256. The $2 \times 2 \times 256$ tensor is then flattered to a 1024-dim latent vector.

The PointNet branch encoder contains four residual blocks, and each are with two fully connected layers with residual connections. At the beginning of each block we concatenate the roots' coordinates in UV and 3D spaces. At the end of each block we apply a max pooling aggregation and concatenated to the feature of each point cloud, as the input to the next layer, until reaching the end of the fourth block where the max-pooled feature is the global feature from the PointNet branch. The latent dimensions in these blocks are 32,32,64, and 128.

We concatenate the 1024-dim feature from the conv branch and the 128-dim feature, and map them to mean and standard deviation vectors of 1024-dim to reparameterise the latent code of the VAE. The decoder is the reserves of the encoder and similarly the depth-to-space upsampling design [37] in adapted in 2D convs.

In between each two adjacent linear or convolution layers throughout the model, we use Swish activation [29] and Group Normalization [44] which divides latent channels into 8 groups.

**Densification model**   The two-layer MLP in the message-passing graph convolution has latent dimension of 32 The number of neighbouring guide features used in the graph aggregation is 8.

**High-frequency VAE**   The architecture is mostly the same as the guide curve VAE except that some layers need to change the input dimension after the guide VAE latent code is concatenated. In addition, the sequence length of $\tilde{l}^H$ is 32 in the frequency space so there are one more downsampling block in the 1D conv encoder and one more upsampling block in the 1D decoder.

**Loss functions**   The Variational Autoencoder (VAE) framework employs the Evidence Lower Bound (ELBO) which comprises two principal components: the KL divergence term and the reconstruction

term. For our low-frequency guide curve model, the reconstruction term is quantified using the mean squared error (MSE) L2 loss on the signal, supplemented by an additional L2 loss on the signal's derivative. Furthermore, a specific term is dedicated to optimising the reconstruction of the density map, as detailed in Section 3. In the densification process, we introduce a set of losses for the original signal, its derivative, and the density map associated with dense strands. These losses are integrated with a discounting weight of 0.1 to balance their contribution. For the high-frequency model, we incorporate a KL divergence term for the high-frequency latent variable, alongside an L1 loss for the accurate reconstruction of the Discrete Cosine Transform (DCT) harmonics.

**Density map sampling**   At the end of the conv branch in the dual-branch decoder, we output a density map, same to the spatial resolution of the convolution feature map. The density map is trained to optimise towards the ground truth probability values of a root points falling in each of the grid, for each training example. In training, we use oracle root points from the encoder and optimise the density map and strands simultaneously. During inference, we can sample root points from the probability maps when the oracle root points from the encoder are unknown, before generating strand details. Each sampled root points is assigned with a random 2D UV coordinate within the small square grid it comes from. In practice, we output two density maps from the convolution branch decoder, one for guide and one for dense strands.

Although the density map sampling does not guarantee the exact same original root positions, empirically with a reasonable resolution of density map, we find that the resulting root positions correctly resemble the distribution of root points. Our evaluation are all based on set chamfer measurements, thus not requiring strand-to-strand correspondence for evaluation.

**Evaluation of upsampling and the nearest neighbour baseline**   For both our learning-based upsampling and nearest neighbour upsampling, we first need to sample a collection of root points for the dense strands on the head scalp UV map. In the usual pipeline of VAE generation and reconstruction, this is achieved by sampling from the learned density map. Specifically in the experimental evaluation here, we use oracle roots from the ground truth to ensure fair and direct comparisons. In the nearest neighbour upsampling, for each sampled root of the dense strand, we identify one guide strand whose root is the nearest to the sampled root. Then we make the dense strand at this root is the same as the nearest guide strand, i.e., we copy this guide strand and translate it to the sampled root.

**Evaluation metric**   We use aggregated squared Euclidean distance $d(l, l') = \frac{1}{n}\|l - l'\|_2^2 := \frac{1}{n}\sum_{t=0}^{n-1} \|l(t) - l'(t)\|_2^2$ to evaluate distance of two individual strand curves, and the bidirectional chamfer distance based on the individual strand metric to evaluate distance of two hairstyles as sets of strands: $\frac{1}{|\mathcal{H}|} \sum_{l \in \mathcal{H}} \min_{l' \in \mathcal{H'}} d(l, l') + \frac{1}{|\mathcal{H'}|} \sum_{l' \in \mathcal{H'}} \min_{l \in \mathcal{H}} d(l', l)$.

# E  Data

The source of synthetic hairstyle data is twofolds. We use 343 synthetic hairstyles that originate from mesh hair cards [15], thus consisting of smooth strands with straight and wavy structures in low frequencies without high-frequency details. In addition, we collected and crafted 26 base particle hair projects using Blender [8] in the form of genuine 3D strands. The 343 hair mesh converted hair and the 26 base strand hair are used for training the low-pass signals in the guide curve VAE and the densification model. For high-frequency signals, the 26 base particle hair are further augmented into 315 examples by operating with the frequency and the magnitude of curliness, and adding random high-frequency noise in Blender. These augmented data are used to train conditional VAE model for high-frequency signal completion, while those from the mesh hair cards are precisely enough to be represented with the first eight DCT harmonics.

# F  Limitations

The quality of generated hair is constrained by the scarcity of high-quality, strand-based hair datasets. Even with heavy data augmentation as proposed in [48, 41] for local visual features, the variability

of the rough geometry of the principal growth directions indicated by the guide curves is limited. Therefore, acquiring a diverse collection of high-quality strand hair examples is crucial for learning a robust generative latent manifold that generates more creative hairstyles.

A widespread limitation of the task of strand hair generation is lack of suitable evaluation metrics. quantitative evaluation of hair generation is hard, because currently there is no such a measurement to evaluate the generation quality of strand hair. Evaluation of image generation can take domain-specific PSNR, SSIM measures, as well as FID and LPIPS that require a pretrained semantic encoder (eg. VGGNet). Unfortunately, strand hair has neither these domain-specific measures nor a VGGNet-like semantic encoder for strands to use FID and LPIPS measurements.

Another challenge is the efficiency of our proposed method. As it is a geometric deep learning model that handles sets of permutation-equivariant elements, it does not match the runtime efficiency of purely convolutional architectures. For instance, generating guide strands with a batch size of 32 requires 0.7 seconds, whereas a pure 2D convolutional baseline performs inference in just 0.09 seconds. The complete generation process, including densification and high-frequency detail addition, takes about 12 seconds. Despite being less efficient, the generation time remains within a practical range for real-world applications.

Some occasional failure cases include messy isolated flying strands and penetration into head meshes. as in Fig. 11.

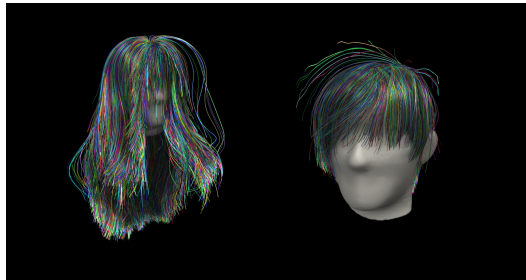

Figure 11: Failure examples.

